# Anatomically Constrained Decoding of Finger Flexion from Electrocorticographic Signals

**Zuoguan Wang**
Department of ECSE
Rensselaer Polytechnic Inst.
Troy, NY 12180
wangz6@rpi.edu

**Gerwin Schalk**
Wadsworth Center
NYS Dept of Health
Albany, NY, 12201
schalk@wadsworth.org

**Qiang Ji**
Department of ECSE
Rensselaer Polytechnic Inst.
Troy, NY 12180
jiq@rpi.edu

## Abstract

Brain-computer interfaces (BCIs) use brain signals to convey a user's intent. Some BCI approaches begin by decoding kinematic parameters of movements from brain signals, and then proceed to using these signals, in absence of movements, to allow a user to control an output. Recent results have shown that electrocorticographic (ECoG) recordings from the surface of the brain in humans can give information about kinematic parameters (e.g., hand velocity or finger flexion). The decoding approaches in these demonstrations usually employed classical classification/regression algorithms that derive a linear mapping between brain signals and outputs. However, they typically only incorporate little prior information about the target kinematic parameter. In this paper, we show that different types of anatomical constraints that govern finger flexion can be exploited in this context. Specifically, we incorporate these constraints in the construction, structure, and the probabilistic functions of a switched non-parametric dynamic system (SNDS) model. We then apply the resulting SNDS decoder to infer the flexion of individual fingers from the same ECoG dataset used in a recent study. Our results show that the application of the proposed model, which incorporates anatomical constraints, improves decoding performance compared to the results in the previous work. Thus, the results presented in this paper may ultimately lead to neurally controlled hand prostheses with full fine-grained finger articulation.

## 1   Introduction

Brain computer interfaces (BCIs) allow people to control devices directly using brain signals [19]. Because BCI systems directly convert brain signals into commands to control output devices, they can be used by people with severe paralysis. Core components of any BCI system are the feature extraction algorithm that extracts those brain signal features that represent the subject's intent, and the decoding algorithm that translates those features into output commands to control artificial actuators.

Substantial efforts in signal processing and machine learning have been devoted to decoding algorithms. Many of these efforts focused on classifying discrete brain states. The linear and non-linear classification algorithms used in these efforts are reviewed in [12, 1, 10]. The simplest translation algorithms use linear models to model the relationship between brain signals and limb movements. This linear relationship can be defined using different algorithms, including multiple linear regression, pace regression [8], or ridge regression [13]. Other studies have explored the use of non-linear methods, including neural networks [15], multilinear perceptrons [7], and support vector machines [7]. Despite substantial efforts, it is still unclear whether non-linear methods can provide consistent benefits over linear methods in the BCI context.

What is common to current linear and non-linear methods is that they are often used to model the instantaneous relationship between brain signals and particular behavioral parameters. Thus, they

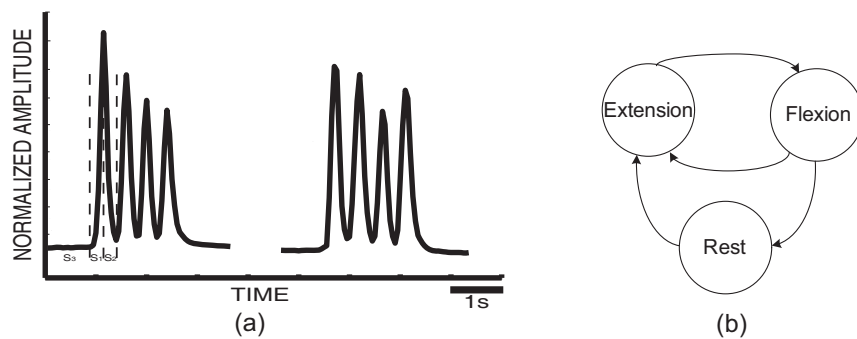

Figure 1: (a) Examples of two flexion traces. (b) A diagram of possible state transitions for finger movements.

do not account for the temporal evolution of movement parameters, and can also not directly provide uncertainty in their predictions. Furthermore, existing methods do not offer opportunities to incorporate prior knowledge about the target model system. In the example of finger flexion, existing methods cannot readily account for the physiological, physical, and mechanical constraints that affect the flexion of different fingers. The main question we sought to answer with this study is whether mathematical decoding algorithms that can make use of the temporal evolution of movement parameters, that can incorporate uncertainty, and that can also incorporate prior knowledge, would provide improved decoding results compared to an existing algorithm that utilized only the instantaneous relationship.

Some previous studies evaluated models that can utilize temporal evolutions. These include the Kalman filter (KF) that explicitly characterizes the temporal evolution of movement parameters [20]. One important benefit offered by Kalman filters (KFs) is that as a probabilistic method, it can provide confidence estimates for its results. Hidden Markov Models (HMMs) represent another dynamic model that can allow to model the latent space both spatially and temporally. As a generalization of HMMs and KFs, switching linear dynamic systems (SLDSs) provide more expressive power for sequential data. Standard SLDS has also been used in BCI research, where it was used for inference of hand motion from motor cortical neurons [21]. Apart from its expressive power, as a probabilistic graphical model, SLDS has a flexible structural framework that facilitates the incorporation of prior knowledge by specifying parameters or structures. Nevertheless, no previous study has evaluated a method that can utilize temporal evolutions, incorporate uncertainty, and make use of different types of constraints.

The proposed SNDS addresses several limitations of SLDS in terms of modeling the anatomical constraints of the finger flexion. We applied the SNDS technique to a dataset used in previous studies ([8]) to decode from ECoG signals the flexion of individual fingers, and we compared decoding results when we did and did not use anatomical constraints (i.e., for SNDS/regression and regression). Our results show that incorporation of anatomical constraints substantially improved decoding results compared to when we did not incorporate this information. We attribute this improvement to the following technical advances. First, and most importantly, we introduce a prior model based on SNDS, which takes advantage of anatomical constraints about finger flexion. Second, to effectively model the duration of movement patterns, our model solves the "Markov assumption" problem more efficiently by modeling the dependence of state transition on the continuous state variable. Third, because estimation of continuous transition is crucial to accurate prediction, we applied kernel density estimation to model the continuous state transition. Finally, we developed effective learning and inference methods for the SNDS model.

## 2   Modeling of Finger Flexion

Figure 1 (a) shows two examples for typical flexion traces. From this figure, we can make the following observations:

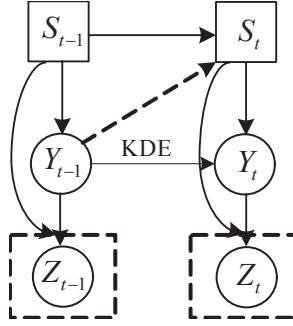

Figure 2: SNDS model in which $S_t$, $Y_t$, $Z_t$ represent the moving states, real finger position and the measurement of finger position at time $t$ respectively.

1. The movement of fingers can be categorized into three states: extension (state $S_1$), flexion (state $S_2$) and rest (rest state $S_3$).
2. For each state, there are particular predominant movement patterns. In the extension state $S_1$, the finger keeps moving away from the rest position. In the flexion state $S_2$, the finger moves back to the rest position. In the rest state $S_3$, there are only very small movements.
3. For either state $S_1$ or state $S_2$, the movement speed is relatively low toward full flexion or full extension, but faster in between. For the rest state, the speed stays close to zero.
4. The natural flexion or extension of fingers are limited to certain ranges due to the physical constraints of our hand.
5. The transition between different states is not random. Figure 1 (b) shows the four possible transitions between three states. The extension state and flexion state can transfer to each other, while the rest state can only follow the flexion state and can only precede the extension state. This is also easy to understand from our common sense about natural finger flexion. When the finger is extended, it is impossible for it to directly transition into the rest state without experiencing flexion first. Similarly, fingers can not transition from rest state to flexion state without first going through the extension state.
6. Figure 1 (b) discusses four possible ways of state transitions. The probability of these transitions depends on the finger position. For example, in the situation at hand, it is unlikely that the extension state transfers to the flexion state right after the extension state begins. At the same time, it is more likely to occur when the finger has extended enough and is near the end. Similar situations occur at other state transitions.

In summary, the observations described above provide constraints that govern finger flexion patterns. Using the methods described below, we will build a computational model that incorporates these constraints and that can systematically learn the movement patterns from data.

## 3 Model Construction

In this section, we show how the constraints summarized above are incorporated into the construction of the finger flexion model. The overall structure of our model is shown in Figure 2. The top layer S represents moving states that include the extension state ($S_1$), flexion state ($S_2$), and rest state ($S_3$). The middle layer (continuous state variable) represents the real finger position, and the bottom layer (observation) $Z$ the measurements of finger positions. We discuss each layer in detail below.

### 3.1 State Transitions

In the standard SLDS, the probability of duration $\tau$ of state $i$ is, according to the Markov assumption, defined as follows:

$$P(\tau) = q_{ii}^{\tau}(1 - q_{ii}) \tag{1}$$

where $q_{ii}$ denotes the transition probability of state $i$ when it makes a self transition. Equation 1 states that the probability of staying in a given state decreases exponentially with time. This behavior can not provide an adequate representation for many natural temporal phenomena. The natural finger

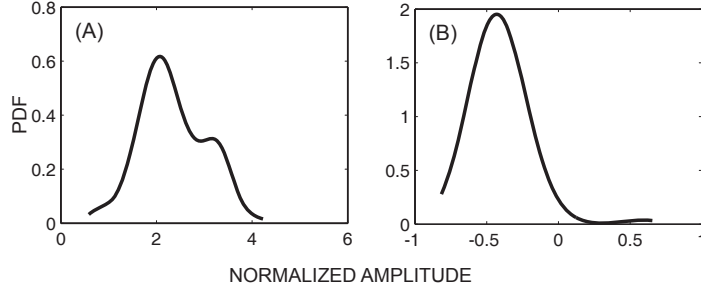

Figure 3: (a) Probabilistic density function (PDF) of $Y_{t-1}$ given $S_{t-1} = extension$ and $S_t = flexion$; (b) Probabilistic density function of $Y_{t-1}$ given $S_{t-1} = flexion$ and $S_t = extension$.

flexion is an example. It usually takes a certain amount of time for fingers to finish extension or flexion. Thus, the duration of certain movement patterns will deviate from the distribution described by Equation 1.

This limitation of the state duration model has been investigated by [2, 14]. In fact, in many cases, the temporal variance is dependent on spacial variance, i.e., state transition is dependent on continuous state variables. In the context of finger flexion, as discussed in Section 2, the transition of moving states is dependent on finger position. In the model shown in Figure 2, the variable $S_t$ not only has an incoming arrow from $S_{t-1}$ but also from $Y_{t-1}$:

$$P(S_t|Y_{t-1}, S_{t-1}) = \frac{1}{P(Y_{t-1}, S_{t-1})} P(Y_{t-1}, S_{t-1}, S_t) \tag{2}$$

$$= \frac{P(S_{t-1})}{P(Y_{t-1}, S_{t-1})} P(Y_{t-1}|S_{t-1}, S_t) P(S_t|S_{t-1})$$

$$= \frac{1}{P(Y_{t-1}|S_{t-1})} P(Y_{t-1}|S_{t-1}, S_t) P(S_t|S_{t-1})$$

where $P(Y_{t-1}|S_{t-1})$ is a normalization term with no relation to $S_t$. $P(S_t|S_{t-1})$ is the state transition, which is same with that in HMM and standard SLDS. $P(Y_{t-1}|S_{t-1}, S_t)$ is the posterior probability of $Y_{t-1}$ given state transition from $S_{t-1}$ to $S_t$. $P(Y_{t-1}|S_{t-1}, S_t)$ plays a central role in controlling state transition. It directly relates state transition to finger position. We take the transition between extension state and flexion state as an example to give an intuitive explanation. Figure 3(a) shows that the transition from extension state to flexion state most probably happens at the finger position between 1.5 and 2.5, which is near the extension end of movement. Similarly, Figure 3(b) implies that when the finger position is between -0.6 and -0.3, which is the flexion end of the finger movement, the transition from flexion state to extension state has a high probability.

## 3.2 Continuous State Transition

In SLDSs, the $Y$ transition is linearly modeled. However, in our model, the continuous state transition is still highly nonlinear during the extension and flexion states. This is mainly because the finger movement speed is uneven (fast in the middle but slow at the beginning and end). Modeling the continuous state transition properly is important for accurate decoding of finger movement. Here we propose a nonparametric method with which continuous state transitions are modeled using kernel density estimation [3]. A Gaussian kernel is the most common choice because of its effectiveness and tractability. With a Gaussian kernel, the joint estimated joint distribution $\hat{p}(Y_{t-1}, Y_t)$ under each state can be obtained by:

$$\hat{p}(Y_{t-1} = y_{t-1}, Y_t = y_t) = \frac{1}{Nh_{Y_{t-1}}h_{Y_t}} \sum_{j=1}^{N} K\left(\frac{y_{t-1} - y_{j-1}}{h_{Y_{t-1}}}\right) K\left(\frac{y_t - y_j}{h_{Y_t}}\right). \tag{3}$$

where $K(\cdot)$ is a given kernel function; $h_{Y_{t-1}}$ and $h_{Y_t}$ are numeric bandwidth for $Y_{t-1}$ and $Y_t$. $N$ is the total number of training examples. Our choice for $K(\cdot)$ is a Gaussian kernel $K(t) = (2\pi)^{-1/2}e^{-t^2/2}$. Bandwidths $h_{Y_{t-1}}$ and $h_{Y_t}$ are estimated via a leave-one-out likelihood criterion

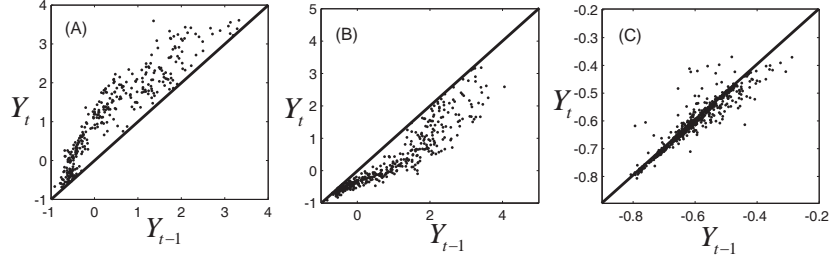

Figure 4: (a) kernel locations for $\hat{p}(Y_{t-1}, Y_t)$ under extension state; (b) kernel locations for $\hat{p}(Y_{t-1}, Y_t)$ under flexion state; kernel locations for $\hat{p}(Y_{t-1}, Y_t)$ under rest state. Numbers on the axis are the normalized amplitude of the fingers' flexion.

[9], which maximizes:

$$LCV(h_{Y_{t-1}}, h_{Y_t}) = \prod_{i=1}^{N} \hat{p}_{\{h_{Y_{t-1}}, h_{Y_t}, -i\}}(y_{i-1}, y_i) \tag{4}$$

where $\hat{p}_{\{h_{Y_{t-1}}, h_{Y_t}, -i\}}(y_{i-1}, y_i)$ denotes the density estimated with $(y_{i-1}, y_i)$ deleted. $\hat{p}(Y_{t-1}, Y_t)$ provides a much more accurate representation of continuous state transition than does a linear model. Figure 4 gives an example of the kernel locations for $\hat{p}(Y_{t-1}, Y_t)$ under each of the three states (trained with part of the data from thumb flexion of subject A). Even though kernel locations do not represent the joint distribution $\hat{p}(Y_{t-1}, Y_t)$, they do help to gain some insight into the relationship between $Y_{t-1}$ and $Y_t$. Each graph in Figure 4 describes a temporal transition pattern for each movement pattern. For the extension state, all kernel locations are above the diagonal, which means that statistically $Y_t$ is greater than $Y_{t-1}$, i.e., fingers are moving up. Also the farther the kernel locations are from the diagonal, the larger the value of $Y_t - Y_{t-1}$, which implies greater moving speed at time $t$. In the extension state, the moving speed around average flexion is statistically greater that around the two extremes (full flexion and extension). Similar arguments can be applied to the flexion state in Figure 4(b). For the rest state, kernel locations are almost along the diagonal, which means $Y_t = Y_{t-1}$, i.e., fingers are not moving. The capability of being able to model the non-linear dependence of speed on position under each state is critical to make a precise prediction of the flexion trace.

### 3.3 Observation Model

$Z$ is the observation which is the finger flexion trace directly mapped from ECoG signals through other regression algorithms. In this paper, we employ the pace regression for this mapping. Here we make an assumption that under each movement pattern, $Z_t$ depends linearly on $Y_t$, and corrupted by Gaussian noise. Specifically, this relationship can be represented by a linear Gaussian [4]:

$$Z_t = \alpha^{(s)} Y_t + w^{(s)}, \qquad w^{(s)} \sim N(\mu, \sigma^{(s)^2}) \tag{5}$$

Parameters $\alpha^{(s)}, \mu^{(s)}$ and $\sigma^{(s)^2}$ can be estimated from the training data via: $\alpha^{(s)} = \frac{E[ZY] - E[Z]E[Y]}{E[Y^2] - E^2[Y]}$, $\mu^{(s)} = E[Z] - \alpha E[Y]$ and $\sigma^{(s)^2} = E[Z^2] - E^2[Z] - \frac{(E[ZY] - E[Z]E[Y])^2}{E[Y^2] - E^2[Y]}$, where $E$ represents the statistical expectation and it is approximated by the sample mean.

### 3.4 Learning and Inference

#### 3.4.1 Learning

All variables of the SNDS model are incorporated during learning. Finger flexion states are estimated from the behavioral flexion traces (e.g., Figure 1 (a)). Specifically, samples on the extension parts of the traces are labeled with state "extension," samples on the flexion parts of the traces are labeled with state "flexion," and samples during rest are labeled with state "rest." $Y$ is the true flexion trace, which we approximate with the data glove measurements. $Z$ is the observation for which we use the output of pace regression.

All parameters $\bar{\bar{\Phi}}$ in our model (Figure 2) consist of three components: the state transition parameter $\bar{\Phi}_S$, continuous state transition parameter $\bar{\Phi}_Y$, and observation parameter $\bar{\Phi}_O$. For state transition parameter $\bar{\Phi}_S$, as discussed in Equation 2, $P(S_t|S_{t-1})$ and $P(Y_{t-1}|S_{t-1}, S_t)$ are learned from the training data. $P(S_t|S_{t-1})$ can be simply obtained by counting. However, here we need to enforce the constraints described in section 2(5). The elements in the conditional probability table of $P(S_t|S_{t-1})$ corresponding to the impossible state transitions are set to zero. $P(Y_{t-1}|S_{t-1}, S_t)$ is estimated by kernel density estimation using the one-dimensional form of Equation 1. $Y$ transition parameter $\bar{\Phi}_Y$ includes the joint distribution $\hat{p}(Y_{t-1}, Y_t)$, which can be estimated using Equation 3 in which bandwidths were selected using the criteria in Equation 4. $\bar{\Phi}_O$ includes $\alpha^{(s)}$, $\mu^{(s)}$ and $\sigma^{(s)2}$ and they can be estimated using Equations in section 3.3.

### 3.4.2 Inference

Given the time course of ECoG signals, our goal is to infer the time course of finger flexion. This is a typical filtering problem, that is, recursively estimating the posterior distribution of $S_t$ and $Y_t$ given the observation from the beginning to time $t$, i.e.,$Z_{1:t}$:

$$P(S_t, Y_t|Z_{1:t}) \propto P(Z_t|S_t, Y_t, Z_{1:t-1})P(S_t, Y_t|Z_{1:t-1}) \quad (6)$$

$$= P(Z_t|S_t, Y_t)\left[\sum_{S_{t-1}}\int_{Y_{t-1}} P(S_t, Y_t|S_{t-1}, Y_{t-1})P(S_{t-1}, Y_{t-1}|Z_{1:t-1})\right]$$

$$= P(Z_t|S_t, Y_t)\left[\sum_{S_{t-1}}\int_{Y_{t-1}} P(S_t|S_{t-1}, Y_{t-1})P(Y_t|S_t, Y_{t-1})P(S_{t-1}, Y_{t-1}|Z_{1:t-1})\right]$$

where $P(S_{t-1}, Y_{t-1}|Z_{1:t-1})$ is the filtering result of the former step. However we note that not all the continuous variables in our model follow Gaussian distribution, because kernel density estimation was used to model the dynamics of the continuous state variable. Hence, it is infeasible to update the posterior distribution $P(S_t, Y_t|Z_{1:t})$ analytically in each step. To cope with this issue, we adopted a numerical sampling method based on particle filtering [6] to propagate and update the discretely approximated distribution over time.

## 4 Experiments

### 4.1 Data Collection

The section gives a brief overview of data collection and feature extraction. A more comprehensive description is given in [8]. The study included five subjects – three women (subjects A, C and E) and two men (subject B and D). Each subject had a 48- or 64-electrode grid placed over the fronto-parietal-temporal region including parts of sensorimotor cortex. During the experiment, the subjects were asked to repeatedly flex and extend specific individual fingers according to visual cues that were given on a video screen. Typically, the subjects flexed the indicated finger 3-5 times over a period of 1.5-3 s and then rested for 2 s. The data collection for each subject lasted 10 min, which yielded an average of 30 trials for each finger. The flexion of each finger was measured by a data glove (5DT Data Glove 5 Ultra, Fifth Dimension Technologies), which digitized the flexion of each finger at 12 bit resolution.

The ECoG signals from the electrode grid were recorded using the general-purpose BCI2000 system [17, 16] connected to a Neuroscan Synamps2 system. All electrodes were referenced to an inactive electrode. The signals were further amplified, bandpass filtered between 0.15 and 200 Hz, and digitized at 1000 Hz. Each dataset was visually inspected and those channels that did not clearly contain ECoG activity were removed, which resulted in 48, 63, 47, 64 and 61 channels (for subjects A-E respectively) for subsequent analyses.

### 4.2 Feature Extraction

Feature extraction was identical to that in [8]. In short, we first re-referenced the signals using a common average reference (CAR), which subtracted $\frac{1}{H}\sum_{q=1}^{H} s_q$ from each channel, where $H$ is the total number of channels and $s_q$ is the collected signal at the $qth$ channel and at the particular

time. For each 100-ms time slice (overlapped by 50 ms) and each channel, we converted these time-series ECoG data into the frequency domain using an autogressive model of order 20 [11]. Using this model, we derived frequency amplitudes between 0 to 500 Hz in 1 Hz bins. ECoG features were extracted by averaging these frequency amplitudes across five frequency ranges, i.e., 8-12 Hz, 18-24 Hz, 75-115 Hz, 125-159 Hz, and 159-175 Hz. In addition to the frequency features described above, we obtained the Local Motor Potential (LMP) [18] by averaging the raw time-domain signal at each channel over 100-ms time window. This resulted in 6 features for each of the ECoG channels, e.g., a total of 288 features from 48 channels.

### 4.3 Evaluation

We defined a movement period as the time between 1000 ms prior to movement onset and 1000 ms after movement offset. Movement onset was defined as the time when the finger's flexion value exceeded an empirically defined threshold. Conversely, movement offset was defined as the time when the finger's flexion value fell below that threshold and no movement onset was detected within the next 1200 ms [8]. To achieve a dataset with relatively balanced movement and rest periods, we discarded all data outside the movement period. For each finger, we used 5-fold cross validation to evaluate the performance of our modeling and inference algorithms that are described in more detail in the following sections, i.e., 4/5th of data was used for training and 1/5th of data was used for testing. Finally, we compared the performance with that achieved using pace regression (which had been used in [8]). To do this, we used the PaceRegression algorithm implemented in the Java-based Weka package [5].

### 4.4 Results

To give an impression of the qualitative improvement of our modeling algorithms described above compared to pace regression, we first provide a qualitative example of the results achieved with each method on the index finger of subject A. These results are shown in Figure 5. In this figure, the top panel shows results achieved using pace regression and the middle figure shows results achieved using SNDS. In each of these two panels, the thin dotted line shows the actual flexion of the index finger (concatenated for five movement periods), and the thick solid line shows the flexion decoded using pace regression/SNDS. This figure demonstrates qualitatively that the decoding of finger flexion achieved using SNDS much better approximates the actual finger flexion than does pace regression. We also observe that SNDS produces much smoother predictions, which is mainly due to the consideration of temporal evolution of movement parameters in SNDS. The bottom panel again shows the actual flexion pattern (thin dotted line) as well as the finger flexion state (1=flexion, 2=extension, 3=rest; thick solid line). These results demonstrate that the state of finger flexion (which cannot be directly inferred using a method that does not incorporate a state machine (such as pace regression)) can be accurately inferred using SNDS. In addition to the qualitative comparison provided above, Table 1 gives a quantitative comparison between the results achieved using SNDS and pace regression. The results presented in this table give mean square errors (MSE) (min/max/mean computed across the cross validation folds). They show that for all fingers and all subjects, the results achieved using SNDS are superior to those achieved using pace regression. The overall average of mean square error reduces from 0.86 (pace regression) to 0.64 (SNDS). This improvement of SNDS compared to pace regression was highly statistically significant: when computed a paired t-test on the mean square errors for all fingers and subjects and between pace regression and SNDS, the resulting $p$-value was $<< 0.001$.

## 5  Discussion

This paper demonstrates that anatomical constraints can be successfully captured to build switched non-parametric dynamic systems to decode finger flexion from ECoG signals. We also showed that the resulting computational models are more accurately able to infer the flexion of individual fingers than does pace regression, an established technique that has recently been used on the same dataset. This improvement is possible by dividing the flexion activity into several moving states ($S_t$), considering the state transition over time, establishing specific state transition by considering its dependence on the finger position (continuous state variable $Y_t$) and modeling the individual transition pattern of continuous state variables under each moving state accurately by using kernel density estimation.

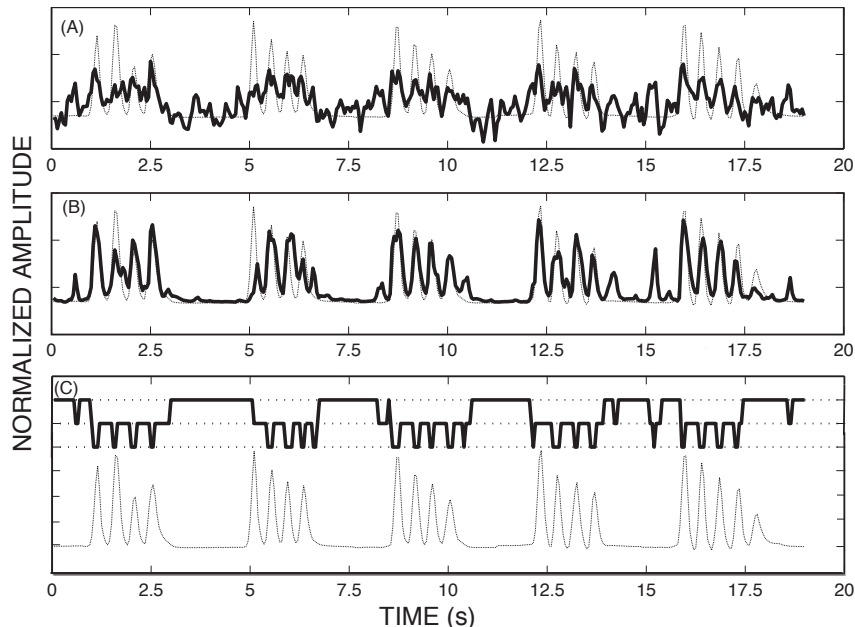

Figure 5: (a) Actual finger flexion (dotted trace) and decoded finger flexion (solid trace) using pace regression (mean square error 0.68); (b) Actual finger flexion (dotted trace) and decoded finger flexion (solid trace) using SNDS (mean square error 0.40); (c) Actual finger flexion (dotted trace) and state prediction (solid trace).

Table 1: **Comparison of decoding performance between pace regression and SNDS.** Results are given, for a particular finger and subject, as mean square errors between actual and decoded movement (minimum, maximum and mean across all cross validation folds).

| Subject | Alg. | Thumb | Index Finger | Middle Finger | Ring Finger | Little Finger | Avg. |
|---|---|---|---|---|---|---|---|
| A | pace | 0.49/0.64/0.58 | 0.61/0.68/0.64 | 0.74/0.84/0.77 | 0.77/0.93/0.86 | 0.74/0.85/0.81 | 0.73 |
| A | SNDS | 0.27/0.45/0.35 | 0.40/0.51/0.44 | 0.57/0.76/0.63 | 0.64/0.86/0.73 | 0.52/0.68/0.59 | 0.54 |
| B | pace | 0.56/0.81/0.65 | 0.46/0.99/0.63 | 0.47/0.87/0.68 | 0.43/0.62/0.52 | 0.46/0.85/0.60 | 0.62 |
| B | SNDS | 0.31/0.62/0.46 | 0.32/0.80/0.44 | 0.32/0.67/0.49 | 0.25/0.50/0.39 | 0.23/0.64/0.40 | 0.43 |
| C | pace | 0.69/1.03/0.83 | 0.73/0.79/0.78 | 0.79/1.07/0.87 | 0.79/1.01/0.89 | 0.81/1.12/0.97 | 0.87 |
| C | SNDS | 0.33/0.85/0.53 | 0.35/0.54/0.46 | 0.48/0.60/0.54 | 0.44/0.76/0.61 | 0.60/0.95/0.73 | 0.56 |
| D | pace | 1.18/1.42/1.29 | 0.82/1.21/1.07 | 0.90/1.05/0.99 | 0.98/1.17/1.09 | 1.17/1.43/1.27 | 1.14 |
| D | SNDS | 0.97/1.28/1.15 | 0.75/1.08/0.94 | 0.82/0.90/0.87 | 0.92/1.04/0.96 | 0.94/1.19/1.0 | 0.98 |
| E | pace | 0.94/1.09/1.03 | 0.76/1.15/0.96 | 0.56/0.98/0.80 | 0.85/1.04/0.94 | 0.71/1.05/0.90 | 0.93 |
| E | SNDS | 0.75/1.01/0.84 | 0.57/1.00/0.75 | 0.44/0.77/0.63 | 0.63/0.82/0.73 | 0.43/0.90/0.68 | 0.71 |

Generally, this improvement in decoding performance is possible, because the computational model puts different types of constraints on the possible flexion predictions. In other words, the model may not be able to produce all possible finger flexion patterns. However, the constraints that we put on these finger flexions are based on the actual natural finger flexions, and thus should not be limiting for other natural flexions of individual fingers. However, to what extent these constraints used here may generalize to those of simultaneous movements of multiple fingers remains to be explored.

There are some directions in which this work could be further improved. First, to reduce the computational complexity caused by kernel density estimation, non-linear transition functions can be used to model the continuous state transitions. Second, more efficient inference methods could be developed to replace standard particle sampling. Finally, the methods presented in this paper could be extended to allow for simultaneous decoding of all five fingers instead of one at a time.

## References

[1] Bashashati, Ali, Fatourechi, Mehrdad, Ward, Rabab K., and Birch, Gary E. A survey of signal processing algorithms in brain-computer interfaces based on electrical brain signals. *J. Neural*

*Eng.*, 4(2):R32+, June 2007. ISSN 1741-2552. doi: 10.1088/1741-2560/4/2/R03.

[2] Ferguson, J. Variable duration models for speech. In *Proc. Symp. on the Application of Hidden Markov Models to Text and Speech*, pp. 143–79, 1980.

[3] Frank, Eibe, Trigg, Leonard, Holmes, Geoffrey, and Witten, Ian H. Naive Bayes for Regression. In *Machine Learning*, pp. 5–26, 1998.

[4] Friedman, Nir, Goldszmidt, Moises, and Lee, Thomas J. Bayesian network classification with continuous attributes: Getting the best of both discretization and parametric fitting. In *ICML*, pp. 179–187. Morgan Kaufmann, 1998.

[5] Hall, Mark, Frank, Eibe, Holmes, Geoffrey, Pfahringer, Bernhard, Reutemann, Peter, and Witten, Ian H. The weka data mining software: an update. *SIGKDD Explor. Newsl.*, 11:10–18, November 2009. ISSN 1931-0145.

[6] Isard, Michael and Blake, Andrew. Condensation - conditional density propagation for visual tracking. *International Journal of Computer Vision*, 29:5–28, 1998.

[7] Kim, Kyung Hwan, Kim, Sung Shin, and Kim, Sung June. Superiority of nonlinear mapping in decoding multiple single-unit neuronal spike trains: A simulation study. *Journal of Neuroscience Methods*, 150(2):202 – 211, 2006. ISSN 0165-0270.

[8] Kubánek, J, Miller, K J, Ojemann, J G, Wolpaw, J R, and Schalk, G. Decoding flexion of individual fingers using electrocorticographic signals in humans. *J Neural Eng*, 6(6):066001–066001, Dec 2009.

[9] Loader, Clive R. Bandwidth Selection: Classical or Plug-In? *The Annals of Statistics*, 27(2): 415–438, 1999. ISSN 00905364. URL http://www.jstor.org/stable/120098.

[10] Lotte, F, Congedo, M, Lécuyer, A, Lamarche, F, and Arnaldi, B. A review of classification algorithms for EEG-based brain-computer interfaces. *J Neural Eng*, 4(2):1–1, Jun 2007.

[11] Marple, S. L. *Digital spectral analysis: with applications*. Prentice-Hall, Inc., Upper Saddle River, NJ, USA, 1986. ISBN 0-132-14149-3.

[12] Muller, K.-R., Anderson, C.W., and Birch, G.E. Linear and nonlinear methods for brain-computer interfaces. *Neural Systems and Rehabilitation Engineering, IEEE Transactions on*, 11(2):165 –169, june 2003. ISSN 1534-4320. doi: 10.1109/TNSRE.2003.814484.

[13] Mulliken, Grant H., Musallam, Sam, and Andersen, Richard A. Decoding Trajectories from Posterior Parietal Cortex Ensembles. *J. Neurosci.*, 28(48):12913–12926, 2008.

[14] Russell, M. and Moore, R. Explicit modelling of state occupancy in hidden Markov models for automatic speech recognition. In *ICASSP*, volume 10, pp. 5–8, Apr 1985.

[15] Sanchez, Justin C., Erdogmus, Deniz, and Principe, Jose C. Comparison between nonlinear mappings and linear state estimation to model the relation from motor cortical neuronal firing to hand movements. In *Proceedings of SAB Workshop*, pp. 59–65, 2002.

[16] Schalk, G and Mellinger, J. *A Practical Guide to Brain-Computer Interfacing with BCI2000*. Springer, 2010.

[17] Schalk, G., McFarland, D. J., Hinterberger, T., Birbaumer, N., and Wolpaw, J. R. BCI2000: a general-purpose brain-computer interface (BCI) system. *Biomedical Engineering, IEEE Transactions on*, 51(6):1034–1043, June 2004.

[18] Schalk, G, Kubánek, J, Miller, K J, Anderson, N R, Leuthardt, E C, Ojemann, J G, Limbrick, D, Moran, D, Gerhardt, L A, and Wolpaw, J R. Decoding two-dimensional movement trajectories using electrocorticographic signals in humans. *J Neural Eng*, 4(3):264–75, Sep 2007.

[19] Wolpaw, Jonathan R. Brain-computer interfaces (BCIs) for communication and control. In *ACM SIGACCESS*, Assets '07, pp. 1–2. ACM, 2007.

[20] Wu, Wei, Black, Michael J., Gao, Yun, Bienenstock, Elie, Serruya, Mijail, Shaikhouni, Ali, and Donoghue, John P. Neural decoding of cursor motion using a Kalman filter, 2003.

[21] Wu, Wei, Black, M.J., Mumford, D., Gao, Yun, Bienenstock, E., and Donoghue, J.P. A switching Kalman filter model for the motor cortical coding of hand motion. In *IEMBS*, volume 3, pp. 2083 – 2086 Vol.3, sept. 2003. doi: 10.1109/IEMBS.2003.1280147.

